# Exact MAP Estimates by (Hyper)tree Agreement

**Martin J. Wainwright,**
Department of EECS,
UC Berkeley,
Berkeley, CA 94720
martinw@eecs.berkeley.edu

**Tommi S. Jaakkola and Alan S. Willsky,**
Department of EECS,
Massachusetts Institute of Technology,
Cambridge, MA, 02139
{tommi,willsky}@mit.edu

## Abstract

We describe a method for computing provably exact *maximum a posteriori* (MAP) estimates for a subclass of problems on graphs with cycles. The basic idea is to represent the original problem on the graph with cycles as a convex combination of tree-structured problems. A convexity argument then guarantees that the optimal value of the original problem (i.e., the log probability of the MAP assignment) is upper bounded by the combined optimal values of the tree problems. We prove that this upper bound is met with equality if and only if the tree problems share an optimal configuration in common. An important implication is that any such shared configuration must also be the MAP configuration for the original problem. Next we develop a tree-reweighted max-product algorithm for attempting to find convex combinations of tree-structured problems that share a common optimum. We give necessary and sufficient conditions for a fixed point to yield the exact MAP estimate. An attractive feature of our analysis is that it generalizes naturally to convex combinations of hypertree-structured distributions.

## 1 Introduction

Integer programming problems arise in various fields, including machine learning, statistical physics, communication theory, and error-correcting coding. In many cases, such problems can be formulated in terms of undirected graphical models [e.g., 1], in which the cost function corresponds to a graph-structured probability distribution, and the problem of interest is to find the *maximum a posteriori* (MAP) configuration.

In previous work [2], we have shown how to use convex combinations of tree-structured distributions in order to upper bound the log partition function. In this paper, we apply similar ideas to upper bound the log probability of the MAP configuration. As we show, this upper bound is met with equality whenever there is a configuration that is optimal for all trees, in which case it must also be a MAP configuration for the original problem. The work described here also makes connections with the max-product algorithm [e.g., 3, 4, 5], a well-known method for attempting to compute the MAP configuration, one which is exact for trees but approximate for graphs with cycles. In the context of coding problems, Frey and Koetter [4] developed an attenuated version of max-product, which is guaranteed to find the MAP codeword if it converges. One contribution of this paper is to develop a *tree-reweighted max-product algorithm* that attempts to find a collection of

tree-structured problems that share a common optimum. This algorithm, though similar to both the standard and attenuated max-product updates [4], differs in key ways.

The remainder of this paper is organized as follows. The next two subsections provide background on exponential families and convex combinations. In Section 2, we introduce the basic form of the upper bounds on the log probability of the MAP assignment, and then develop necessary and sufficient conditions for it to tight (i.e., met with equality). In Section 3, we develop tree-reweighted max-product algorithms for attempting to find a convex combination of trees that yields a tight bound. We prove that for positive compatibility functions, the algorithm always has at least one fixed point; moreover, if a key uniqueness condition is satisfied, the configuration specified by a fixed point must be MAP optimal. We also illustrate how the algorithm, like the standard max-product algorithm [5], can fail if the uniqueness condition is not satisfied. We conclude in Section 4 with pointers to related work, and extensions of the current work.

## 1.1 Notation and set-up

Consider an undirected (simple) graph $G = (V, E)$. For each vertex $s \in V$, let $x_s$ be a random variable taking values in the discrete space $\mathcal{X}_s = \{0, 1, \ldots, m_s - 1\}$. We use the letters $j, k$ to denote particular elements of the sample space $\mathcal{X}_s$. The overall random vector $\mathbf{x} = \{x_s \mid s \in V\}$ takes values in the Cartesian product space $\mathcal{X}^N = \mathcal{X}_1 \times \cdots \times \mathcal{X}_N$, where $N = |V|$. We make use of the following exponential representation of a graph-structured distribution $p(\mathbf{x})$. For some index set $\mathcal{I}$, we let $\boldsymbol{\phi} = \{\phi_\alpha \mid \alpha \in \mathcal{I}\}$ denote a collection of potential functions defined on the cliques of $G$, and let $\theta = \{\theta_\alpha \mid \alpha \in \mathcal{I}\}$ be a vector of real-valued weights on these potential functions. The exponential family determined by $\boldsymbol{\phi}$ is the collection of distributions $p(\mathbf{x}; \theta) \propto \exp\{\sum_{\alpha \in \mathcal{I}} \theta_\alpha \phi_\alpha(\mathbf{x})\}$.

In a *minimal* exponential representation, the functions $\{\phi_\alpha\}$ are affinely independent. For example, one minimal representation of a binary process (i.e., $\mathcal{X}_s = \{0, 1\}$ for all $s$) using pairwise potential functions is the usual Ising model, in which the collection of potentials $\boldsymbol{\phi} = \{x_s \mid s \in V\} \cup \{x_s x_t \mid (s, t) \in E\}$. In this case, the index set is given by $\mathcal{I} = V \cup E$. In most of our analysis, we use an *overcomplete* representation, in which there are linear dependencies among the potentials $\{\phi_\alpha\}$. In particular, we use indicator functions as potentials:

$$
\begin{aligned}
\phi_{s;j}(x_s) &= \delta_{s;j}(x_s), \quad s \in V; \; j \in \mathcal{X}_s & \text{(1a)} \\
\phi_{st;jk}(x_s, x_t) &= \delta_{s;j}(x_s)\delta_{t;k}(x_t), \; (s, t) \in E; \; (j, k) \in \mathcal{X}_s \times \mathcal{X}_t & \text{(1b)}
\end{aligned}
$$

where the indicator function $\delta_{s;j}(x_s)$ is equal to one if $x_s = j$, and zero otherwise. In this case, the index set $\mathcal{I}$ consists of the union of $\mathcal{I}(V) = \{(s; j) \mid s \in V; \; j \in \mathcal{X}_s\}$ with the edge indices $\mathcal{I}(E) = \{(st; jk) \mid (s, t) \in E; \; (j, k) \in \mathcal{X}_s \times \mathcal{X}_t\}$.

Of interest to us is the *maximum a posteriori* configuration $\widehat{\mathbf{x}}_{\mathrm{MAP}} = \arg \max_{\mathbf{x} \in \mathcal{X}^N} p(\mathbf{x}; \theta)$. Equivalently, we can express this MAP configuration as the solution of the integer program $\mathcal{F}(\theta) = \max_{\mathbf{x} \in \mathcal{X}^N} J(\mathbf{x}; \theta)$, where

$$
J(\mathbf{x}; \theta) = \langle \theta, \boldsymbol{\phi}(\mathbf{x}) \rangle = \sum_{s \in V} \sum_j \theta_{s;j} \phi_{s;j}(x_s) + \sum_{(s,t) \in E} \sum_{j,k} \theta_{st;jk} \phi_{st;jk}(x_s, x_t) \quad \text{(2)}
$$

Note that the function $\mathcal{F}(\theta)$ is the maximum of a collection of linear functions, and hence is convex [6] as a function of $\theta$, which is a key property for our subsequent development.

## 1.2 Convex combinations of trees

Let $\widehat{\theta}$ be a particular parameter vector for which we are interested in computing $\mathcal{F}(\widehat{\theta})$. In this section, we show how to derive upper bounds via the convexity of $\mathcal{F}$. Let $\mathcal{T}$ denote

a particular spanning tree of $G$, and let $\mathfrak{T} = \mathfrak{T}(G)$ denote the set of all spanning trees. For each spanning tree $\mathcal{T} \in \mathfrak{T}$, let $\theta(\mathcal{T})$ be an exponential parameter vector of the same dimension as $\theta$ that respects the structure of $\mathcal{T}$. To be explicit, if $\mathcal{T}$ is defined by an edge set $E(\mathcal{T}) \subset E$, then $\theta(\mathcal{T})$ must have zeros in all elements corresponding to edges not in $E(\mathcal{T})$. However, given an edge $e$ belonging to two trees $\mathcal{T}_1$ and $\mathcal{T}_2$, the quantity $\theta_e(\mathcal{T}_1)$ can be different than $\theta_e(\mathcal{T}_2)$. For compactness, let $\Theta := \{\theta(\mathcal{T}) \mid \mathcal{T} \in \mathfrak{T}\}$ denote the full collection, where the notation $\theta(\mathcal{T})$ specifies those subelements of $\Theta$ corresponding to spanning tree $\mathcal{T}$.

In order to define a convex combination, we require a probability distribution $\vec{\mu}$ over the set of spanning trees — that is, a vector $\vec{\mu} := \{\mu(\mathcal{T}), \mathcal{T} \in \mathfrak{T} \mid \mu(\mathcal{T}) \geq 0\}$ such that $\sum_{\mathcal{T} \in \mathfrak{T}} \mu(\mathcal{T}) = 1$. For any distribution $\vec{\mu}$, we define its *support*, denoted by $\operatorname{supp}(\vec{\mu})$, to be the set of trees to which it assigns strictly positive probability. In the sequel, we will also be interested in the probability $\mu_e = \operatorname{Pr}_{\vec{\mu}}\{e \in \mathcal{T}\}$ that a given edge $e \in E$ appears in a spanning tree $\mathcal{T}$ chosen randomly under $\vec{\mu}$. We let $\boldsymbol{\mu_e} = \{\mu_e \mid e \in E\}$ represent a vector of *edge appearance probabilities*, which must belong to the spanning tree polytope [see 2]. We say that a distribution $\vec{\mu}$ (or the vector $\boldsymbol{\mu_e}$) is *valid* if $\mu_e > 0$ for every edge $e \in E$.

A *convex combination* of exponential parameter vectors is defined via the weighted sum $\sum_{\mathcal{T} \in \mathfrak{T}} \mu(\mathcal{T})\theta(\mathcal{T})$, which we denote compactly as $\mathbb{E}_{\vec{\mu}}[\theta(\mathcal{T})]$. Of particular importance are collections of exponential parameters $\Theta$ for which there exists a convex combination that is equal to $\widehat{\theta}$. Accordingly, we define the set $\mathcal{A}(\widehat{\theta}) := \{(\Theta; \vec{\mu}) \mid \mathbb{E}_{\vec{\mu}}[\theta(\mathcal{T})] = \widehat{\theta}\}$. For any valid distribution $\vec{\mu}$, it can be seen that there exist pairs $(\Theta; \vec{\mu}) \in \mathcal{A}(\widehat{\theta})$.

**Example 1 (Single cycle).** To illustrate these definitions, consider a binary distribution ($\mathcal{X}_s = \{0, 1\}$ for all nodes $s \in V$) defined by a single cycle on 4 nodes. Consider a target distribution in the minimal Ising form $p(\mathbf{x}; \widehat{\theta}) = \exp\{x_1 x_2 + x_2 x_3 + x_3 x_4 + x_4 x_1 - \Phi(\widehat{\theta})\}$; otherwise stated, the target distribution is specified by the minimal parameter $\widehat{\theta} = [0\ 0\ 0\ 0\ \ 1\ 1\ 1\ 1]$, where the zeros represent the fact that $\widehat{\theta}_s = 0$ for all $s \in V$. The

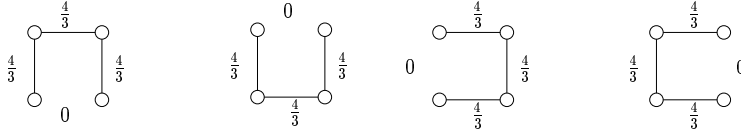

**Figure 1.** A convex combination of four distributions $p(\mathbf{x}; \theta(\mathcal{T}_i))$, each defined by a spanning tree $\mathcal{T}_i$, is used to approximate the target distribution $p(\mathbf{x}; \widehat{\theta})$ on the single-cycle graph.

four possible spanning trees $\mathfrak{T} = \{\mathcal{T}_i \mid i = 1, \dots, 4\}$ on a single cycle on four nodes are illustrated in Figure 1. We define a set of associated exponential parameters $\Theta = \{\theta(\mathcal{T}_i)\}$ as follows:

$$\theta(\mathcal{T}_1) = \frac{4}{3}[0\ 0\ 0\ 0\ 1\ 1\ 1\ 0] \qquad \theta(\mathcal{T}_3) = \frac{4}{3}[0\ 0\ 0\ 0\ 1\ 0\ 1\ 1]$$

$$\theta(\mathcal{T}_2) = \frac{4}{3}[0\ 0\ 0\ 0\ 1\ 1\ 0\ 1] \qquad \theta(\mathcal{T}_4) = \frac{4}{3}[0\ 0\ 0\ 0\ 0\ 1\ 1\ 1]$$

Finally, we choose $\mu(\mathcal{T}_i) = 1/4$ for all $\mathcal{T}_i \in \mathfrak{T}$. With this uniform distribution over trees, we have $\mu_e = 3/4$ for each edge, and $\mathbb{E}_{\vec{\mu}}[\theta(\mathcal{T})] = \widehat{\theta}$, so that $(\Theta; \vec{\mu}) \in \mathcal{A}(\widehat{\theta})$.

## 2 Optimal upper bounds

With the set-up of the previous section, the basic form of the upper bounds follows by applying Jensen's inequality [6]. In particular, for any pair $(\Theta; \vec{\mu}) \in \mathcal{A}(\widehat{\theta})$, we have the upper bound $\mathcal{F}(\widehat{\theta}) \leq \mathbb{E}_{\vec{\mu}}[\mathcal{F}(\theta(\mathcal{T}))]$. The goal of this section is to examine this bound, and understand when it is met with equality. In more explicit terms, the upper bound can be written as:

$$\mathcal{F}(\widehat{\theta}) \leq \sum_{\mathcal{T}} \mu(\mathcal{T})\mathcal{F}(\theta(\mathcal{T})) = \sum_{\mathcal{T}} \mu(\mathcal{T}) \max_{\mathbf{x} \in \mathcal{X}^N} \langle \theta(\mathcal{T}), \phi(\mathbf{x}) \rangle \tag{3}$$

Now suppose that there exists an $\widehat{\mathbf{x}} \in \mathcal{X}^N$ that attains the maximum defining $\mathcal{F}(\theta(\mathcal{T}))$ for each tree $\mathcal{T} \in \mathrm{supp}(\vec{\mu})$. In this case, it is clear that the bound (3) is met with equality. An important implication is that the configuration $\widehat{\mathbf{x}}$ also attains the maximum defining $\mathcal{F}(\widehat{\theta})$, so that it is an *optimal* solution to the original problem.

In fact, as we show below, the converse to this statement also holds. More formally, for any exponential parameter vector $\theta(\mathcal{T})$, let $\mathrm{OPT}(\theta(\mathcal{T}))$ be the collection of configurations $\mathbf{x}$ that attain the maximum defining $\mathcal{F}(\theta(\mathcal{T}))$, defined as follows:

$$\mathrm{OPT}(\theta(\mathcal{T})) = \{\mathbf{x} \in \mathcal{X}^N \mid \langle \theta(\mathcal{T}), \phi(\mathbf{x}') \rangle \leq \langle \theta(\mathcal{T}), \phi(\mathbf{x}) \rangle \text{ for all } \mathbf{x}' \in \mathcal{X}^N\} \tag{4}$$

With this notation, the critical property is that the intersection $\mathrm{OPT}(\Theta) := \cap_{\mathcal{T}} \mathrm{OPT}(\theta(\mathcal{T}))$ of configurations optimal for all tree-structured problems is non-empty. We thus have the following result:

**Proposition 1 (Tightness of bound).** *The bound of equation* (3) *is tight if and only if there exists a configuration $\widehat{\mathbf{x}} \in \mathcal{X}^N$ that for each $\mathcal{T} \in \mathrm{supp}(\vec{\mu})$ achieves the maximum defining $\mathcal{F}(\theta(\mathcal{T}))$. In other words, $\widehat{\mathbf{x}} \in \mathrm{OPT}(\Theta)$.*

*Proof.* Consider some pair $(\Theta; \vec{\mu}) \in \mathcal{A}(\widehat{\theta})$. Let $\widehat{\mathbf{x}}$ be a configuration that attains the maximum defining $\mathcal{F}(\widehat{\theta})$. We write the difference of the RHS and the LHS of equation (3) as follows:

$$0 \leq \left[\sum_{\mathcal{T}} \mu(\mathcal{T})\mathcal{F}(\theta(\mathcal{T}))\right] - \mathcal{F}(\widehat{\theta}) = \left[\sum_{\mathcal{T}} \mu(\mathcal{T})\mathcal{F}(\theta(\mathcal{T}))\right] - \langle \widehat{\theta}, \phi(\widehat{\mathbf{x}}) \rangle$$

$$= \sum_{\mathcal{T}} \mu(\mathcal{T})\left[\mathcal{F}(\theta(\mathcal{T})) - \langle \theta(\mathcal{T}), \phi(\widehat{\mathbf{x}}) \rangle\right]$$

Now for each $\mathcal{T} \in \mathrm{supp}(\vec{\mu})$, the term $\mathcal{F}(\theta(\mathcal{T})) - \langle \theta(\mathcal{T}), \phi(\widehat{\mathbf{x}}) \rangle$ is non-negative, and equal to zero only when $\widehat{\mathbf{x}}$ belongs to $\mathrm{OPT}(\theta(\mathcal{T}))$. Therefore, the bound is met with equality if and only if $\widehat{\mathbf{x}}$ achieves the maximum defining $\mathcal{F}(\theta(\mathcal{T}))$ for all trees $\mathcal{T} \in \mathrm{supp}(\vec{\mu})$. $\square$

Proposition 1 motivates the following strategy: given a spanning tree distribution $\vec{\mu}$, find a collection of exponential parameters $\Theta^* = \{\theta^*(\mathcal{T})\}$ such that the following holds: (a) Admissibility: The pair $(\Theta^*; \vec{\mu})$ satisfies $\sum_{\mathcal{T}} \mu(\mathcal{T})\theta^*(\mathcal{T}) = \widehat{\theta}$. (b) Mutual agreement: The intersection $\cap_{\mathcal{T}} \mathrm{OPT}(\theta^*(\mathcal{T}))$ of tree-optimal configurations is non-empty.

If (for a fixed $\vec{\mu}$) we are able to find a collection $\Theta^*$ satisfying these two properties, then Proposition 1 guarantees that all configurations in the (non-empty) intersection $\cap_{\mathcal{T}} \mathrm{OPT}(\theta^*(\mathcal{T}))$ achieve the maximum defining $\mathcal{F}(\widehat{\theta})$. As discussed above, assuming that $\vec{\mu}$ assigns strictly positive probability to every edge in the graph, satisfying the admissibility condition is not difficult. It is the second condition of mutual optimality on all trees that poses the challenge.

# 3 Mutual agreement via equal max-marginals

We now develop an algorithm that attempts to find, for a given spanning tree distribution $\vec{\mu}$, a collection $\Theta^* = \{\theta^*(\mathcal{T})\}$ satisfying both of these properties. Interestingly, this algorithm is related to the ordinary max-product algorithm [3, 5], but differs in several key ways. While this algorithm can be formulated in terms of reparameterization [e.g., 5], here we present a set of message-passing updates.

## 3.1 Max-marginals

The foundation of our development is the fact [1] that any tree-structured distribution $p(\mathbf{x}; \theta(\mathcal{T}))$ can be factored in terms of its max-marginals. In particular, for each node $s \in V$, the corresponding single node max-marginal is defined as follows:

$$T_s(x_s) \quad = \quad \max_{\{\mathbf{x}' \mid x_s' = x_s\}} p(\mathbf{x}'; \theta(\mathcal{T})) \tag{5}$$

In words, for each $x_s \in \mathcal{X}_s$, $T_s(x_s)$ is the maximum probability over the subset of configurations $\mathbf{x}'$ with element $x_s'$ fixed to $x_s$. For each edge $(s, t) \in E$, the pairwise max-marginal is defined analogously as $T_{st}(x_s, x_t) = \max_{\{\mathbf{x}' \mid (x_s', x_t') = (x_s, x_t)\}} p(\mathbf{x}'; \theta(\mathcal{T}))$. With these definitions, the max-marginal tree factorization [1] is given by:

$$p(\mathbf{x}; \theta(\mathcal{T})) \quad \propto \quad \prod_{s \in V} T_s(x_s) \prod_{(s,t) \in E(\mathcal{T})} \frac{T_{st}(x_s, x_t)}{T_s(x_s) T_t(x_t)} \tag{6}$$

One interpretation of the ordinary max-product algorithm for trees, as shown in our related work [5], is as computing this alternative representation.

Suppose moreover that for each node $s \in V$, the following uniqueness condition holds:
Uniqueness Condition: *For each $s \in V$, the max-marginal $T_s$ has a unique optimum $x_s^*$.*
In this case, the vector $\mathbf{x}^* = \{x_s^* \mid s \in V\}$ is the MAP configuration for the tree-structured distribution [see 5].

## 3.2 Tree-reweighted max-product

The tree-reweighted max-product method is a message-passing algorithm, with fixed points that specify a collection of tree exponential parameters $\Theta^* = \{\theta^*(\mathcal{T})\}$ satisfying the admissibility condition. The defining feature of $\Theta^*$ is that the associated tree distributions $p(\mathbf{x}; \theta^*(\mathcal{T}))$ all share a common set $\mathbf{T}^* = \{T_s^*, T_{st}^*\}$ of max-marginals. In particular, for a given tree $\mathcal{T}$ with edge set $E(\mathcal{T})$, the distribution $p(\mathbf{x}; \theta^*(\mathcal{T}))$ is specified compactly by the subcollection $\Pi^{\mathcal{T}}(\mathbf{T}^*) := \{T_s^* \mid s \in V\} \cup \{T_{st}^* \mid (s, t) \in E(\mathcal{T})\}$ as follows:

$$p(\mathbf{x}; \theta^*(\mathcal{T})) \equiv p^{\mathcal{T}}(\mathbf{x}; \mathbf{T}^*) \quad := \quad \kappa \prod_{s \in V} T_s^*(x_s) \prod_{(s,t) \in E(\mathcal{T})} \frac{T_{st}^*(x_s, x_t)}{T_s^*(x_s) T_t^*(x_t)} \tag{7}$$

where $\kappa$ is a constant[1] independent of $\mathbf{x}$. As long as $\mathbf{T}^*$ satisfies the Uniqueness Condition, the configuration $\mathbf{x}^* = \{x_s^* \mid s \in V\}$ must be the MAP configuration for each tree-structured distribution $p(\mathbf{x}; \theta^*(\mathcal{T}))$. This mutual agreement on trees, in conjunction with the admissibility of $\Theta^*$, implies that $\mathbf{x}^*$ is also the MAP configuration for $p(\mathbf{x}; \widehat{\theta})$.

For each valid $\boldsymbol{\mu}_e$, there exists a tree-reweighted max-product algorithm designed to find the requisite set $\mathbf{T}^*$ of max-marginals via a sequence of message-passing operations. For each edge $(s, t) \in E$, let $M_{ts}(x_s)$ be the message passed from node $t$ to node $s$. It is a vector of length $m_s$, with one element for each state $j \in \mathcal{X}_s$. We use $\phi_s(x_s; \widehat{\theta})$ as a

shorthand for $\sum_j \widehat{\theta}_{s;j} \phi_{s;j}(x_s)$, with the quantity $\phi_{st}(x_s, x_t; \widehat{\theta}_{st})$ similarly defined. We use the messages $\mathbf{M} = \{M_{st}\}$ to specify a set of functions $\mathbf{T} = \{T_s, T_{st}\}$ as follows:

$$T_s(x_s) = \exp\left(\phi_s(x_s; \widehat{\theta}_s)\right) \prod_{u \in \Gamma(s)} \left[M_{us}(x_s)\right]^{\mu_{us}} \tag{8a}$$

$$T_{st}(x_s, x_t) = \varphi_{st}(x_s, x_t; \widehat{\theta}) \frac{\prod_{u \in \Gamma(s) \backslash t} \left[M_{us}(x_s)\right]^{\mu_{us}}}{\left[M_{ts}(x_s)\right]^{(1 - \mu_{st})}} \frac{\prod_{u \in \Gamma(t) \backslash s} \left[M_{ut}(x_t)\right]^{\mu_{ut}}}{\left[M_{st}(x_t)\right]^{(1 - \mu_{ts})}} \tag{8b}$$

where $\varphi_{st}(x_s, x_t; \widehat{\theta}) = \exp\left(\frac{1}{\mu_{st}} \phi_{st}(x_s, x_t; \widehat{\theta}_{st}) + \phi(x_s; \widehat{\theta}_s) + \widehat{\theta}_t \phi(x_t; \widehat{\theta}_t)\right)$.

For each tree $\mathcal{T}$, the subcollection $\Pi^{\mathcal{T}}(\mathbf{T})$ can be used to define a tree-structured distribution $p^{\mathcal{T}}(\mathbf{x}; \mathbf{T})$, in a manner analogous to equation (7). By expanding the expectation $\mathbb{E}_{\vec{\mu}}[\log p^{\mathcal{T}}(\mathbf{x}; \mathbf{T})]$ and making use of the definitions of $T_s^*$ and $T_{st}^*$, we can prove the following:

**Lemma 1 (Admissibility).** *Given any collection $\{T_s, T_{st}\}$ defined by a set of messages $\mathbf{M}$ as in equations (8a) and (8b), the convex combination $\sum_{\mathcal{T}} \mu(\mathcal{T}) \log p^{\mathcal{T}}(\mathbf{x}; \mathbf{T})$ is equivalent to $\log p(\mathbf{x}; \widehat{\theta})$ up to an additive constant.*

We now need to ensure that $\mathbf{T} = \{T_s, T_{st}\}$ are a consistent set of max-marginals for each tree-distribution $p^{\mathcal{T}}(\mathbf{x}; \mathbf{T})$. It is sufficient [1, 5] to impose, for each edge $(s, t)$, the *edgewise consistency* condition $\max_{x'_t \in \mathcal{X}_t} T_{st}(x_s, x'_t) = \kappa \, T_s(x_s)$. In order to enforce this condition, we update the messages in the following manner:

---

**Algorithm 1 (Tree reweighted max-product).**
1. Initialize the messages $\mathbf{M}^0 = \{M_{st}^0\}$ with arbitrary positive real numbers.

2. For iterations $n = 0, 1, 2, \ldots$, update the messages as follows:

$$M_{ts}^{n+1}(x_s) = \max_{x'_t \in \mathcal{X}_t} \left\{ \exp\left(\frac{1}{\mu_{st}} \phi_{st}(x_s, x'_t; \widehat{\theta}_{st}) + \phi_t(x'_t; \widehat{\theta}_t)\right) \frac{\prod_{u \in \Gamma(t) \backslash s} \left[M_{ut}^n(x'_t)\right]^{\mu_{ut}}}{\left[M_{st}^n(x'_t)\right]^{(1 - \mu_{ts})}} \right\} \tag{9}$$

---

Using the definitions of $T_s^*$ and $T_{st}^*$, as well as the message update equation (9), the following result can be proved:

**Lemma 2 (Edgewise consistency).** *Let $\mathbf{M}^*$ be a fixed point of the message update equation (9), and let $\mathbf{T}^* = \{T_s^*, T_{st}^*\}$ be defined via $\mathbf{M}^*$ as in equations (8a) and (8b) respectively. Then the edgewise consistency condition is satisfied.*

The message update equation (9) is similar to the standard max-product algorithm [3, 5]. Indeed, if $G$ is actually a tree, then we must have $\mu_{st} = 1$ for every edge $(s, t) \in E$, in which case equation (9) is precisely equivalent to the ordinary max-product update. However, if $G$ has cycles, then it is impossible to have $\mu_{st} = 1$ for every edge $(s, t) \in E$, so that the updates in equation (9) differ from ordinary max-product in some key ways. First of all, the weight $\widehat{\theta}_{st}$ on the potential function $\phi_{st}$ is scaled by the (inverse of the) edge appearance probability $1/\mu_{st} \geq 1$. Secondly, for each neighbor $u \in \Gamma(t) \backslash s$, the incoming message $M_{ut}$ is scaled by the corresponding edge appearance probability $\mu_{ut} \leq 1$. Third of all, in sharp contrast to standard [3] and attenuated [4] max-product updates, the update of message $M_{ts}$ — that is, from $t$ to $s$ along edge $(s, t)$ — depends on the *reverse direction* message $M_{st}$ from $s$ to $t$ along the same edge. Despite these differences, the messages can be updated synchronously as in ordinary max-product. It also possible to perform reparameterization updates over spanning trees, analogous to but distinct from those for ordinary max-product [5]. Such tree-based updates can be terminated once the trees agree on a common configuration, which may happen prior to message convergence [7].

### 3.3 Analysis of fixed points

In related work [5], we established the existence of fixed points for the ordinary max-product algorithm for positive compatibility functions on an arbitrary graph. The same proof can be adapted to show that the tree-reweighted max-product algorithm also has at least one fixed point $\mathbf{M}^*$. Any such fixed point $\mathbf{M}^*$ defines pseudo-max-marginals $\mathbf{T}^*$ via equations (8a) and (8b), which (by design of the algorithm) have the following property:

**Theorem 1 (Exact MAP).** *If $\mathbf{T}^*$ satisfies the Uniqueness Condition, then the configuration $\mathbf{x}^*$ with elements $x_s^* = \arg\max_{x_s' \in \mathcal{X}_s} T_s^*(x_s')$ is a MAP configuration for $p(\mathbf{x}; \widehat{\theta})$.*

*Proof.* For each spanning tree $\mathcal{T} = (V, E(\mathcal{T}))$, the fixed point $\mathbf{T}^*$ defines a tree-structured distribution $p(\mathbf{x}; \theta^*(\mathcal{T}))$ via equation (7). By Lemma 2, the elements of $\mathbf{T}^*$ are edgewise consistent. By the equivalence of edgewise and global consistency for trees [1], the subcollection $\Pi^{\mathcal{T}}(\mathbf{T}^*) = \{T_s^* \mid s \in V\} \cup \{T_{st}^* \mid (s,t) \in E(\mathcal{T})\}$ are exact max-marginals for the tree-structured distribution $p(\mathbf{x}; \theta^*(\mathcal{T}))$. As a consequence, the configuration $\mathbf{x}^*$ must belong to $\mathrm{OPT}(\theta^*(\mathcal{T}))$ for each tree $\mathcal{T}$, so that mutual agreement is satisfied. By Lemma 1, the convex combination $\mathbb{E}_{\vec{\mu}}[\log p(\mathbf{x}; \theta^*(\mathcal{T}))]$ is equal to $\log p(\mathbf{x}; \widehat{\theta})$, so that admissibility is satisfied. Proposition 1 then implies that $\mathbf{x}^*$ is a MAP configuration for $p(\mathbf{x}; \widehat{\theta})$. $\square$

### 3.4 Failures of tree-reweighted max-product

In all of our experiments so far, the message updates of equation (9), if suitably relaxed, have always converged.[2] Rather than convergence problems, the breakdown of the algorithm appears to stem primarily from failure of the Uniqueness Condition. If this assumption is not satisfied, we are no longer guaranteed that the mutual agreement condition is satisfied (i.e., $\mathrm{OPT}(\Theta^*)$ may be empty). Indeed, a configuration $\mathbf{x}^*$ belongs to $\mathrm{OPT}(\Theta^*)$ if and only if the following conditions hold:

Node optimality: *The element $x_s^*$ must achieve $\max_{x_s'} T_s^*(x_s')$ for every $s \in V$.*

Edge optimality: *The pair $(x_s^*, x_t^*)$ must achieve $\max_{(x_s', x_t')} T_{st}^*(x_s', x_t')$ for all $(s,t) \in E$.*

For a given fixed point $\mathbf{T}^*$ that fails the Uniqueness Condition, it may or may not be possible to satisfy these conditions, as the following example illustrates.

**Example 2.** Consider the single cycle on three vertices, as illustrated in Figure 2. We define a distribution $p(\mathbf{x}; \widehat{\theta})$ in an indirect manner, by first defining a set of pseudo-max-marginals $\mathbf{T}^*$ in panel (a). Here $\beta \in [0, 1]$ is a parameter to be specified. Observe that the symmetry of this construction ensures that $\mathbf{T}^*$ satisfies the edgewise consistency condition (Lemma 2) for any $\beta \in [0, 1]$. For each of the three spanning trees of this graph, the collection $\mathbf{T}^*$ defines a tree-structured distribution $p^{\mathcal{T}}(\mathbf{x}; \mathbf{T}^*)$ as in equation (7). We define the underlying distribution via $\log p(\mathbf{x}; \widehat{\theta}) = \mathbb{E}_{\vec{\mu}}[\log p^{\mathcal{T}}(\mathbf{x}; \mathbf{T}^*)] + C$, where $\vec{\mu}$ is the uniform distribution (weight $(1/3)$ on each tree).

In the case $\beta > 0.5$, illustrated in panel (b), it can be seen that two configurations — namely $[0\ 0\ 0]$ and $[1\ 1\ 1]$ — satisfy the node and edgewise optimality conditions. Therefore, each of these configurations are global maxima for the cost function $\mathbb{E}_{\vec{\mu}}[\log p(\mathbf{x}; \mathbf{T}^*)]$. On the other hand, when $\beta < 0.5$, as illustrated in panel (c), any configuration $\mathbf{x}^*$ that is edgewise optimal for all three edges must satisfy $x_s^* \neq x_t^*$ for all $(s,t) \in E$. This is clearly impossible, so that the fixed point $\mathbf{T}^*$ cannot be used to specify a MAP assignment.

Of course, it should be recognized that this example was contrived to break down the algorithm. It should also be noted that, as shown in our related work [5], the standard max-

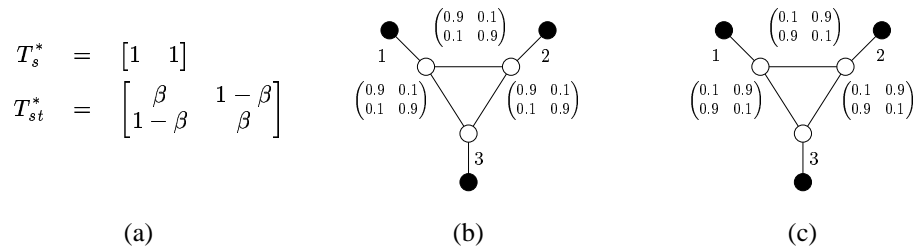

$$T_s^* = \begin{bmatrix} 1 & 1 \end{bmatrix}$$

$$T_{st}^* = \begin{bmatrix} \beta & 1-\beta \\ 1-\beta & \beta \end{bmatrix}$$

(a)             (b)             (c)

**Figure 2.** Cases where the Uniqueness Condition fails. (a) Specification of pseudo-max-marginals $\mathbf{T}^*$. (b) For $\beta > 0.5$, both $[0\ 0\ 0]$ and $[1\ 1\ 1]$ are node and edgewise optimal. (c) For $\beta < 0.5$, no configurations are node and edgewise optimal on the full graph.

product algorithm can also break down when this Uniqueness Condition is not satisfied.

## 4 Discussion

This paper demonstrated the utility of convex combinations of tree-structured distributions in upper bounding the log probability of the MAP configuration. We developed a family of tree-reweighted max-product algorithms for computing optimal upper bounds. In certain cases, the optimal upper bound is met with equality, and hence yields an exact MAP configuration for the original problem on the graph with cycles. An important open question is to characterize the range of problems for which the upper bound is tight. For problems involving a binary-valued random vector, we have isolated a class of problems for which the upper bound is guaranteed to be tight. We have also investigated the Lagrangian dual associated with the upper bound (3). The dual has a natural interpretation as a tree-relaxed linear program, and has been applied to turbo decoding [7]. Finally, the analysis and upper bounds of this paper can be extended in a straightforward manner to hypertrees of of higher width. In this context, hypertree-reweighted forms of generalized max-product updates [see 5] can again be used to find optimal upper bounds, which (when they are tight) again yield exact MAP configurations.

## Footnotes

[1]We use this notation throughout the paper, where the value of $\kappa$ may change from line to line.

[2]In a relaxed message update, we take an $\alpha$-step towards the new (log) message, where $\alpha \in (0, 1]$ is the step size parameter. To date, we have not been able to prove that relaxed updates will always converge.

## References

[1] R. G. Cowell, A. P. Dawid, S. L. Lauritzen, and D. J. Spiegelhalter. *Probablistic Networks and Expert Systems*. Statistics for Engineering and Information Science. Springer-Verlag, 1999.

[2] M. J. Wainwright, T. S. Jaakkola, and A. S. Willsky. A new class of upper bounds on the log partition function. In *Proc. Uncertainty in Artificial Intelligence*, volume 18, pages 536–543, August 2002.

[3] W. T. Freeman and Y. Weiss. On the optimality of solutions of the max-product belief propagation algorithm in arbitrary graphs. *IEEE Trans. Info. Theory*, 47:736–744, 2001.

[4] B. J. Frey and R. Koetter. Exact inference using the attenuated max-product algorithm. In *Advanced mean field methods: Theory and Practice*. MIT Press, 2000.

[5] M. J. Wainwright, T. S. Jaakkola, and A. S. Willsky. Tree consistency and bounds on the max-product algorithm and its generalizations. LIDS Tech. report P-2554, MIT; Available online at http://www.eecs.berkeley.edu/~martinw, July 2002.

[6] D.P. Bertsekas. *Nonlinear programming*. Athena Scientific, Belmont, MA, 1995.

[7] J. Feldman, M. J. Wainwright, and D. R. Karger. Linear programming-based decoding and its relation to iterative approaches. In *Proc. Allerton Conf. Comm. Control and Computing*, October 2002.
